# A Novel Gaussian Sum Smoother for Approximate Inference in Switching Linear Dynamical Systems

**David Barber** and **Bertrand Mesot**
IDIAP Research Institute
Martigny 1920, Switzerland
`david.barber/bertrand.mesot@idiap.ch`

## Abstract

We introduce a method for approximate smoothed inference in a class of switching linear dynamical systems, based on a novel form of Gaussian Sum smoother. This class includes the switching Kalman Filter and the more general case of switch transitions dependent on the continuous latent state. The method improves on the standard Kim smoothing approach by dispensing with one of the key approximations, thus making fuller use of the available future information. Whilst the only central assumption required is projection to a mixture of Gaussians, we show that an additional conditional independence assumption results in a simpler but stable and accurate alternative. Unlike the alternative unstable Expectation Propagation procedure, our method consists only of a single forward and backward pass and is reminiscent of the standard smoothing 'correction' recursions in the simpler linear dynamical system. The algorithm performs well on both toy experiments and in a large scale application to noise robust speech recognition.

## 1 Switching Linear Dynamical System

The Linear Dynamical System (LDS) [1] is a key temporal model in which a latent linear process generates the observed series. For complex time-series which are not well described globally by a single LDS, we may break the time-series into segments, each modeled by a potentially different LDS. This is the basis for the Switching LDS (SLDS) [2, 3, 4, 5] where, for each time $t$, a switch variable $s_t \in 1, \ldots, S$ describes which of the LDSs is to be used. The observation (or 'visible') $v_t \in \mathcal{R}^V$ is linearly related to the hidden state $h_t \in \mathcal{R}^H$ with additive noise $\eta$ by

$$v_t = B(s_t)h_t + \eta^v(s_t) \quad \equiv \quad p(v_t|h_t, s_t) = \mathcal{N}\left(B(s_t)h_t, \Sigma^v(s_t)\right) \tag{1}$$

where $\mathcal{N}(\mu, \Sigma)$ denotes a Gaussian distribution with mean $\mu$ and covariance $\Sigma$. The transition dynamics of the continuous hidden state $h_t$ is linear,

$$h_t = A(s_t)h_{t-1} + \eta^h(s_t), \quad \equiv \quad p(h_t|h_{t-1}, s_t) = \mathcal{N}\left(A(s_t)h_{t-1}, \Sigma^h(s_t)\right) \tag{2}$$

The switch $s_t$ may depend on both the previous $s_{t-1}$ and $h_{t-1}$. This is an augmented SLDS (aSLDS), and defines the model

$$p(v_{1:T}, h_{1:T}, s_{1:T}) = \prod_{t=1}^{T} p(v_t|h_t, s_t)p(h_t|h_{t-1}, s_t)p(s_t|h_{t-1}, s_{t-1})$$

The standard SLDS[4] considers only switch transitions $p(s_t|s_{t-1})$. At time $t = 1$, $p(s_1|h_0, s_0)$ simply denotes the prior $p(s_1)$, and $p(h_1|h_0, s_1)$ denotes $p(h_1|s_1)$.

The aim of this article is to address how to perform inference in the aSLDS. In particular we desire the *filtered* estimate $p(h_t, s_t|v_{1:t})$ and the *smoothed* estimate $p(h_t, s_t|v_{1:T})$, for any $1 \le t \le T$. Both filtered and smoothed inference in the SLDS is intractable, scaling exponentially with time [4].

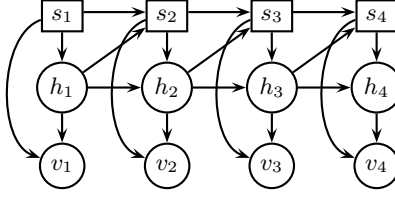

Figure 1: The independence structure of the aSLDS. Square nodes denote discrete variables, round nodes continuous variables. In the SLDS links from $h$ to $s$ are not normally considered.

## 2 Expectation Correction

Our approach to approximate $p(h_t, s_t|v_{1:T})$ mirrors the Rauch-Tung-Striebel 'correction' smoother for the simpler LDS [1].The method consists of a single forward pass to recursively find the filtered posterior $p(h_t, s_t|v_{1:t})$, followed by a single backward pass to correct this into a smoothed posterior $p(h_t, s_t|v_{1:T})$. The forward pass we use is equivalent to standard Assumed Density Filtering (ADF) [6]. The main contribution of this paper is a novel form of backward pass, based only on collapsing the smoothed posterior to a mixture of Gaussians. Together with the ADF forward pass, we call the method Expectation Correction, since it corrects the moments found from the forward pass. A more detailed description of the method, including pseudocode, is given in [7].

### 2.1 Forward Pass (Filtering)

Readers familiar with ADF may wish to continue directly to Section (2.2). Our aim is to form a recursion for $p(s_t, h_t|v_{1:t})$, based on a Gaussian mixture approximation of $p(h_t|s_t, v_{1:t})$. Without loss of generality, we may decompose the filtered posterior as

$$p(h_t, s_t|v_{1:t}) = p(h_t|s_t, v_{1:t})p(s_t|v_{1:t}) \qquad (3)$$

The exact representation of $p(h_t|s_t, v_{1:t})$ is a mixture with $O(S^t)$ components. We therefore approximate this with a smaller $I$-component mixture

$$p(h_t|s_t, v_{1:t}) \approx \sum_{i_t=1}^{I} p(h_t|i_t, s_t, v_{1:t})p(i_t|s_t, v_{1:t})$$

where $p(h_t|i_t, s_t, v_{1:t})$ is a Gaussian parameterized with mean $f(i_t, s_t)$ and covariance $F(i_t, s_t)$. To find a recursion for these parameters, consider

$$p(h_{t+1}|s_{t+1}, v_{1:t+1}) = \sum_{s_t, i_t} p(h_{t+1}|s_t, i_t, s_{t+1}, v_{1:t+1})p(s_t, i_t|s_{t+1}, v_{1:t+1}) \qquad (4)$$

**Evaluating** $p(h_{t+1}|s_t, i_t, s_{t+1}, v_{1:t+1})$

We find $p(h_{t+1}|s_t, i_t, s_{t+1}, v_{1:t+1})$ by first computing the joint distribution $p(h_{t+1}, v_{t+1}|s_t, i_t, s_{t+1}, v_{1:t})$, which is a Gaussian with covariance and mean elements,

$$\Sigma_{hh} = A(s_{t+1})F(i_t, s_t)A^{\mathsf{T}}(s_{t+1}) + \Sigma^h(s_{t+1}), \quad \Sigma_{vv} = B(s_{t+1})\Sigma_{hh}B^{\mathsf{T}}(s_{t+1}) + \Sigma^v(s_{t+1})$$
$$\Sigma_{vh} = B(s_{t+1})F(i_t, s_t), \quad \mu_v = B(s_{t+1})A(s_{t+1})f(i_t, s_t), \quad \mu_h = A(s_{t+1})f(i_t, s_t) \qquad (5)$$

and then conditioning on $v_{t+1}$[1]. For the case $S = 1$, this forms the usual Kalman Filter recursions[1].

**Evaluating** $p(s_t, i_t|s_{t+1}, v_{1:t+1})$

The mixture weight in (4) can be found from the decomposition

$$p(s_t, i_t|s_{t+1}, v_{1:t+1}) \propto p(v_{t+1}|i_t, s_t, s_{t+1}, v_{1:t})p(s_{t+1}|i_t, s_t, v_{1:t})p(i_t|s_t, v_{1:t})p(s_t|v_{1:t}) \quad (6)$$

The first factor in (6), $p(v_{t+1}|i_t, s_t, s_{t+1}, v_{1:t})$ is a Gaussian with mean $\mu_v$ and covariance $\Sigma_{vv}$, as given in (5). The last two factors $p(i_t|s_t, v_{1:t})$ and $p(s_t|v_{1:t})$ are given from the previous iteration. Finally, $p(s_{t+1}|i_t, s_t, v_{1:t})$ is found from

$$p(s_{t+1}|i_t, s_t, v_{1:t}) = \langle p(s_{t+1}|h_t, s_t) \rangle_{p(h_t|i_t, s_t, v_{1:t})} \tag{7}$$

where $\langle \cdot \rangle_p$ denotes expectation with respect to $p$. In the SLDS, (7) is replaced by the Markov transition $p(s_{t+1}|s_t)$. In the aSLDS, however, (7) will generally need to be computed numerically.

**Closing the recursion**

We are now in a position to calculate (4). For each setting of the variable $s_{t+1}$, we have a mixture of $I \times S$ Gaussians which we numerically collapse back to $I$ Gaussians to form

$$p(h_{t+1}|s_{t+1}, v_{1:t+1}) \approx \sum_{i_{t+1}=1}^{I} p(h_{t+1}|i_{t+1}, s_{t+1}, v_{1:t+1})p(i_{t+1}|s_{t+1}, v_{1:t+1})$$

Any method of choice may be supplied to collapse a mixture to a smaller mixture; our code simply repeatedly merges low-weight components. In this way the new mixture coefficients $p(i_{t+1}|s_{t+1}, v_{1:t+1})$, $i_{t+1} \in 1, \ldots, I$ are defined, completing the description of how to form a recursion for $p(h_{t+1}|s_{t+1}, v_{1:t+1})$ in (3). A recursion for the switch variable is given by

$$p(s_{t+1}|v_{1:t+1}) \propto \sum_{s_t, i_t} p(v_{t+1}|s_{t+1}, i_t, s_t, v_{1:t})p(s_{t+1}|i_t, s_t, v_{1:t})p(i_t|s_t, v_{1:t})p(s_t|v_{1:t})$$

where all terms have been computed during the recursion for $p(h_{t+1}|s_{t+1}, v_{1:t+1})$.

The likelihood $p(v_{1:T})$ may be found by recursing $p(v_{1:t+1}) = p(v_{t+1}|v_{1:t})p(v_{1:t})$, where

$$p(v_{t+1}|v_t) = \sum_{i_t, s_t, s_{t+1}} p(v_{t+1}|i_t, s_t, s_{t+1}, v_{1:t})p(s_{t+1}|i_t, s_t, v_{1:t})p(i_t|s_t, v_{1:t})p(s_t|v_{1:t})$$

## 2.2 Backward Pass (Smoothing)

The main contribution of this paper is to find a suitable way to 'correct' the filtered posterior $p(s_t, h_t|v_{1:t})$ obtained from the forward pass into a smoothed posterior $p(s_t, h_t|v_{1:T})$. We derive this for the case of a single Gaussian representation. The extension to the mixture case is straightforward and presented in [7]. We approximate the smoothed posterior $p(h_t|s_t, v_{1:T})$ by a Gaussian with mean $g(s_t)$ and covariance $G(s_t)$ and our aim is to find a recursion for these parameters. A useful starting point for a recursion is:

$$p(h_t, s_t|v_{1:T}) = \sum_{s_{t+1}} p(s_{t+1}|v_{1:T})p(h_t|s_t, s_{t+1}, v_{1:T})p(s_t|s_{t+1}, v_{1:T})$$

The term $p(h_t|s_t, s_{t+1}, v_{1:T})$ may be computed as

$$p(h_t|s_t, s_{t+1}, v_{1:T}) = \int_{h_{t+1}} p(h_t|h_{t+1}, s_t, s_{t+1}, v_{1:t})p(h_{t+1}|s_t, s_{t+1}, v_{1:T}) \tag{8}$$

The recursion therefore requires $p(h_{t+1}|s_t, s_{t+1}, v_{1:T})$, which we can write as

$$p(h_{t+1}|s_t, s_{t+1}, v_{1:T}) \propto p(h_{t+1}|s_{t+1}, v_{1:T})p(s_t|s_{t+1}, h_{t+1}, v_{1:t}) \tag{9}$$

The difficulty here is that the functional form of $p(s_t|s_{t+1}, h_{t+1}, v_{1:t})$ is not squared exponential in $h_{t+1}$, so that $p(h_{t+1}|s_t, s_{t+1}, v_{1:T})$ will not be Gaussian[2]. One possibility would be to approximate the non-Gaussian $p(h_{t+1}|s_t, s_{t+1}, v_{1:T})$ by a Gaussian (or mixture thereof) by minimizing the Kullback-Leilbler divergence between the two, or performing moment matching in the case of a single Gaussian. A simpler alternative (which forms 'standard' EC) is to make the assumption $p(h_{t+1}|s_t, s_{t+1}, v_{1:T}) \approx p(h_{t+1}|s_{t+1}, v_{1:T})$, where $p(h_{t+1}|s_{t+1}, v_{1:T})$ is already known from the previous backward recursion. Under this assumption, the recursion becomes

$$p(h_t, s_t|v_{1:T}) \approx \sum_{s_{t+1}} p(s_{t+1}|v_{1:T})p(s_t|s_{t+1}, v_{1:T}) \langle p(h_t|h_{t+1}, s_t, s_{t+1}, v_{1:t}) \rangle_{p(h_{t+1}|s_{t+1}, v_{1:T})} \tag{10}$$

**Evaluating** $\langle p(h_t|h_{t+1}, s_t, s_{t+1}, v_{1:t}) \rangle_{p(h_{t+1}|s_{t+1}, v_{1:T})}$

$\langle p(h_t|h_{t+1}, s_t, s_{t+1}, v_{1:t}) \rangle_{p(h_{t+1}|s_{t+1}, v_{1:T})}$ is a Gaussian in $h_t$, whose statistics we will now compute. First we find $p(h_t|h_{t+1}, s_t, s_{t+1}, v_{1:t})$ which may be obtained from the joint distribution

$$p(h_t, h_{t+1}|s_t, s_{t+1}, v_{1:t}) = p(h_{t+1}|h_t, s_{t+1})p(h_t|s_t, v_{1:t}) \tag{11}$$

which itself can be found from a forward dynamics from the filtered estimate $p(h_t|s_t, v_{1:t})$. The statistics for the marginal $p(h_t|s_t, s_{t+1}, v_{1:t})$ are simply those of $p(h_t|s_t, v_{1:t})$, since $s_{t+1}$ carries no extra information about $h_t$. The remaining statistics are the mean of $h_{t+1}$, the covariance of $h_{t+1}$ and cross-variance between $h_t$ and $h_{t+1}$, which are given by

$$\langle h_{t+1} \rangle = A(s_{t+1})f_t(s_t), \quad \Sigma_{t+1,t+1} = A(s_{t+1})F_t(s_t)A^\mathsf{T}(s_{t+1}) + \Sigma^h(s_{t+1}), \quad \Sigma_{t+1,t} = A(s_{t+1})F_t(s_t)$$

Given the statistics of (11), we may now condition on $h_{t+1}$ to find $p(h_t|h_{t+1}, s_t, s_{t+1}, v_{1:t})$. Doing so effectively constitutes a reversal of the dynamics,

$$h_t = \overleftarrow{A}(s_t, s_{t+1})h_{t+1} + \overleftarrow{\eta}(s_t, s_{t+1})$$

where $\overleftarrow{A}(s_t, s_{t+1})$ and $\overleftarrow{\eta}(s_t, s_{t+1}) \sim \mathcal{N}(\overleftarrow{m}(s_t, s_{t+1}), \overleftarrow{\Sigma}(s_t, s_{t+1}))$ are easily found using conditioning. Averaging the above reversed dynamics over $p(h_{t+1}|s_{t+1}, v_{1:T})$, we find that $\langle p(h_t|h_{t+1}, s_t, s_{t+1}, v_{1:t}) \rangle_{p(h_{t+1}|s_{t+1}, v_{1:T})}$ is a Gaussian with statistics

$$\mu_t = \overleftarrow{A}(s_t, s_{t+1})g(s_{t+1}) + \overleftarrow{m}(s_t, s_{t+1}), \quad \Sigma_{t,t} = \overleftarrow{A}(s_t, s_{t+1})G(s_{t+1})\overleftarrow{A}^\mathsf{T}(s_t, s_{t+1}) + \overleftarrow{\Sigma}(s_t, s_{t+1})$$

These equations directly mirror the standard RTS backward pass[1].

**Evaluating** $p(s_t|s_{t+1}, v_{1:T})$

The main departure of EC from previous methods is in treating the term

$$p(s_t|s_{t+1}, v_{1:T}) = \langle p(s_t|h_{t+1}, s_{t+1}, v_{1:t}) \rangle_{p(h_{t+1}|s_{t+1}, v_{1:T})} \tag{12}$$

The term $p(s_t|h_{t+1}, s_{t+1}, v_{1:t})$ is given by

$$p(s_t|h_{t+1}, s_{t+1}, v_{1:t}) = \frac{p(h_{t+1}|s_{t+1}, s_t, v_{1:t})p(s_t, s_{t+1}|v_{1:t})}{\sum_{s'_t} p(h_{t+1}|s_{t+1}, s'_t, v_{1:t})p(s'_t, s_{t+1}|v_{1:t})} \tag{13}$$

Here $p(s_t, s_{t+1}|v_{1:t}) = p(s_{t+1}|s_t, v_{1:t})p(s_t|v_{1:t})$, where $p(s_{t+1}|s_t, v_{1:t})$ occurs in the forward pass, (7). In (13), $p(h_{t+1}|s_{t+1}, s_t, v_{1:t})$ is found by marginalizing (11).

Computing the average of (13) with respect to $p(h_{t+1}|s_{t+1}, v_{1:T})$ may be achieved by any numerical integration method desired. A simple approximation is to evaluate the integrand at the mean value of the averaging distribution $p(h_{t+1}|s_{t+1}, v_{1:T})$. More sophisticated methods (see [7]) such as sampling from the Gaussian $p(h_{t+1}|s_{t+1}, v_{1:T})$ have the advantage that covariance information is used[3].

**Closing the Recursion**

We have now computed both the continuous and discrete factors in (8), which we wish to use to write the smoothed estimate in the form $p(h_t, s_t|v_{1:T}) = p(s_t|v_{1:T})p(h_t|s_t, v_{1:T})$. The distribution $p(h_t|s_t, v_{1:T})$ is readily obtained from the joint (8) by conditioning on $s_t$ to form the mixture

$$p(h_t|s_t, v_{1:T}) = \sum_{s_{t+1}} p(s_{t+1}|s_t, v_{1:T})p(h_t|s_t, s_{t+1}, v_{1:T})$$

which may then be collapsed to a single Gaussian (the mixture case is discussed in [7]). The smoothed posterior $p(s_t|v_{1:T})$ is given by

$$p(s_t|v_{1:T}) = \sum_{s_{t+1}} p(s_{t+1}|v_{1:T}) \langle p(s_t|h_{t+1}, s_{t+1}, v_{1:t}) \rangle_{p(h_{t+1}|s_{t+1}, v_{1:T})}. \tag{14}$$

## 2.3 Relation to other methods

The EC Backward pass is closely related to Kim's method [8]. In both EC and Kim's method, the approximation $p(h_{t+1}|s_t, s_{t+1}, v_{1:T}) \approx p(h_{t+1}|s_{t+1}, v_{1:T})$, is used to form a numerically simple backward pass. The other 'approximation' in EC is to numerically compute the average in (14). In Kim's method, however, an update for the discrete variables is formed by replacing the required term in (14) by

$$\langle p(s_t|h_{t+1}, s_{t+1}, v_{1:t}) \rangle_{p(h_{t+1}|s_{t+1}, v_{1:T})} \approx p(s_t|s_{t+1}, v_{1:t}) \tag{15}$$

Since $p(s_t|s_{t+1}, v_{1:t}) \propto p(s_{t+1}|s_t)p(s_t|v_{1:t})/p(s_{t+1}|v_{1:t})$, this can be computed simply from the filtered results alone. The fundamental difference therefore between EC and Kim's method is that the approximation, (15), is not required by EC. The EC backward pass therefore makes fuller use of the future information, resulting in a recursion which intimately couples the continuous and discrete variables. The resulting effect on the quality of the approximation can be profound, as we will see in the experiments.

The Expectation Propagation (EP) algorithm makes the central assumption of collapsing the posteriors to a Gaussian family [5]; the collapse is defined by a consistency criterion on overlapping marginals. In our experiments, we take the approach in [9] of collapsing to a single Gaussian. Ensuring consistency requires frequent translations between moment and canonical parameterizations, which is the origin of potentially severe numerical instability [10]. In contrast, EC works largely with moment parameterizations of Gaussians, for which relatively few numerical difficulties arise. Unlike EP, EC is not based on a consistency criterion and a subtle issue arises about possible inconsistencies in the Forward and Backward approximations for EC. For example, under the conditional independence assumption in the Backward Pass, $p(h_T|s_{T-1}, s_T, v_{1:T}) \approx p(h_T|s_T, v_{1:T})$, which is in contradiction to (5) which states that the approximation to $p(h_T|s_{T-1}, s_T, v_{1:T})$ *will* depend on $s_{T-1}$. Such potential inconsistencies arise because of the approximations made, and should not be considered as separate approximations in themselves. Rather than using a global (consistency) objective, EC attempts to faithfully approximate the exact Forward and Backward propagation routines. For this reason, as in the exact computation, only a single Forward and Backward pass are required in EC.

In [11] a related dynamics reversed is proposed. However, the singularities resulting from incorrectly treating $p(v_{t+1:T}|h_t, s_t)$ as a density are heuristically finessed.

In [12] a variational method approximates the joint distribution $p(h_{1:T}, s_{1:T}|v_{1:T})$ rather than the marginal inference $p(h_t, s_t|v_{1:T})$. This is a disadvantage when compared to other methods that directly approximate the marginal.

Sequential Monte Carlo methods (Particle Filters)[13], are essentially mixture of delta-function approximations. Whilst potentially powerful, these typically suffer in high-dimensional hidden spaces, unless techniques such as Rao-Blackwellization are performed. ADF is generally preferential to Particle Filtering since in ADF the approximation is a mixture of non-trivial distributions, and is therefore more able to represent the posterior.

## 3 Demonstration

Testing EC in a problem with a reasonably long temporal sequence, $T$, is important since numerical instabilities may not be apparent in timeseries of just a few points. To do this, we sequentially generate hidden and visible states from a given model, here with $H = 3$, $S = 2$, $V = 1$ – see Figure(2) for full details of the experimental setup. Then, given only the parameters of the model and the visible observations (but not any of the hidden states $h_{1:T}, s_{1:T}$), the task is to infer $p(h_t|s_t, v_{1:T})$ and $p(s_t|v_{1:T})$. Since the exact computation is exponential in $T$, a simple alternative is to assume that the original sample states $s_{1:T}$ are the 'correct' inferences, and compare how our most probable posterior smoothed estimates $\arg\max_{s_t} p(s_t|v_{1:T})$ compare with the assumed correct sample $s_t$. We chose conditions that, from the viewpoint of classical signal processing, are difficult, with changes in the switches occurring at a much higher rate than the typical frequencies in the signal $v_t$.

For EC we use the mean approximation for the numerical integration of (12). We included the Particle Filter merely for a point of comparison with ADF, since they are not designed to approximate

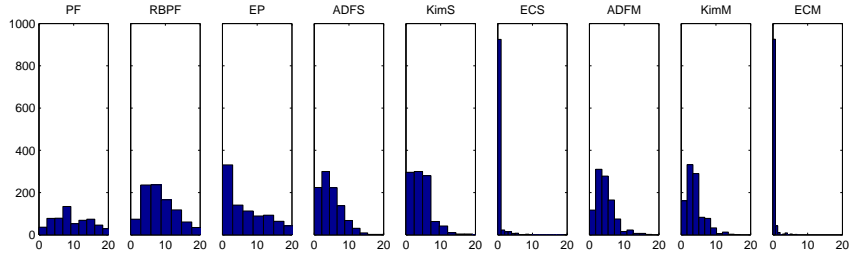

Figure 2: The number of errors in estimating $p(s_t|v_{1:T})$ for a binary switch ($S = 2$) over a time series of length $T = 100$. Hence 50 errors corresponds to random guessing. Plotted are histograms of the errors are over 1000 experiments. The $x$-axes are cut off at 20 errors to improve visualization of the results. (PF) Particle Filter. (RBPF) Rao-Blackwellized PF. (EP) Expectation Propagation. (ADFS) Assumed Density Filtering using a Single Gaussian. (KimS) Kim's smoother using the results from ADFS. (ECS) Expectation Correction using a Single Gaussian ($I = J = 1$). (ADFM) ADF using a multiple of $I = 4$ Gaussians. (KimM) Kim's smoother using the results from ADFM. (ECM) Expectation Correction using a mixture with $I = J = 4$ components. $S = 2$, $V = 1$ (scalar observations), $T = 100$, with zero output bias. $A(s) = 0.9999 * \text{orth}(\text{randn}(\text{H}, \text{H}))$, $B(s) = \text{randn}(\text{V}, \text{H})$. $H = 3$, $\Sigma^h(s) = I_H$, $\Sigma^v(s) = 0.1I_V$, $p(s_{t+1}|s_t) \propto 1_{S \times S} + I_S$. At time $t = 1$, the priors are $p_1 = \text{uniform}$, with $h_1$ drawn from $\mathcal{N}(10 * \text{randn}(\text{H}, 1), I_H)$.

the smoothed estimate, for which 1000 particles were used, with Kitagawa resampling. For the Rao-Blackwellized Particle Filter [13], 500 particles were used, with Kitagawa resampling. We found that EP[4] was numerically unstable and often struggled to converge. To encourage convergence, we used the damping method in [9], performing 20 iterations with a damping factor of 0.5. Nevertheless, the disappointing performance of EP is most likely due to conflicts resulting from numerical instabilities introduced by the frequent conversions between moment and canonical representations.

The best filtered results are given using ADF, since this is better able to represent the variance in the filtered posterior than the sampling methods. Unlike Kim's method, EC makes good use of the future information to clean up the filtered results considerably. One should bear in mind that both EC and Kim's method use the same ADF filtered results. This demonstrates that EC may dramatically improve on Kim's method, so that the small amount of extra work in making a numerical approximation of $p(s_t|s_{t+1}, v_{1:T})$, (12), may bring significant benefits. We found similar conclusions for experiments with an aSLDS[7].

## 4 Application to Noise Robust ASR

Here we briefly present an application of the SLDS to robust Automatic Speech Recognition (ASR), for which the intractable inference is performed by EC, and serves to demonstrate how EC scales well to a large-scale application. Fuller details are given in [14]. The standard approach to noise robust ASR is to provide a set of noise-robust features to a standard Hidden Markov Model (HMM) classifier, which is based on modeling the acoustic feature vector. For example, the method of Unsupervised Spectral Subtraction (USS) [15] provides state-of-the-art performance in this respect. Incorporating noise models directly into such feature-based HMM systems is difficult, mainly because the explicit influence of the noise on the features is poorly understood. An alternative is to model the raw speech signal directly, such as the SAR-HMM model [16] for which, under *clean* conditions, isolated spoken digit recognition performs well. However, the SAR-HMM performs poorly under noisy conditions, since no explicit noise processes are taken into account by the model.

The approach we take here is to extend the SAR-HMM to include an explicit noise process, so that the observed signal $v_t$ is modeled as a noise corrupted version of a clean *hidden* signal $v_t^h$:

$$v_t = v_t^h + \tilde{\eta}_t \quad \text{with} \quad \tilde{\eta}_t \sim \mathcal{N}(0, \tilde{\sigma}^2)$$

| Noise Variance | SNR (dB) | HMM | SAR-HMM | AR-SLDS |
|---|---|---|---|---|
| 0 | 26.5 | 100.0% | 97.0% | 96.8% |
| $10^{-7}$ | 26.3 | 100.0% | 79.8% | 96.8% |
| $10^{-6}$ | 25.1 | 90.9% | 56.7% | 96.4% |
| $10^{-5}$ | 19.7 | 86.4% | 22.2% | 94.8% |
| $10^{-4}$ | 10.6 | 59.1% | 9.7% | 84.0% |
| $10^{-3}$ | 0.7 | 9.1% | 9.1% | 61.2% |

Table 1: Comparison of the recognition accuracy of three models when the test utterances are corrupted by various levels of Gaussian noise.

The dynamics of the clean signal is modeled by a switching AR process

$$v_t^h = \sum_{r=1}^{R} c_r(s_t) v_{t-r}^h + \eta_t^h(s_t), \qquad \eta_t^h(s_t) \sim \mathcal{N}(0, \sigma^2(s_t))$$

where $s_t \in \{1, \ldots, S\}$ denotes which of a set of AR coefficients $c_r(s_t)$ are to be used at time $t$, and $\eta_t^h(s_t)$ is the so-called *innovation* noise. When $\sigma^2(s_t) \equiv 0$, this model reproduces the SAR-HMM of [16], a specially constrained HMM. Hence inference and learning for the SAR-HMM are tractable and straightforward. For the case $\sigma^2(s_t) > 0$ the model can be recast as an SLDS. To do this we define $h_t$ as a vector which contains the $R$ most recent clean hidden samples

$$h_t = \begin{bmatrix} v_t^h & \ldots & v_{t-r+1}^h \end{bmatrix}^\mathsf{T} \tag{16}$$

and we set $A(s_t)$ to be an $R \times R$ matrix where the first row contains the AR coefficients $-c_r(s_t)$ and the rest is a shifted down identity matrix. For example, for a third order ($R = 3$) AR process,

$$A(s_t) = \begin{bmatrix} -c_1(s_t) & -c_2(s_t) & -c_3(s_t) \\ 1 & 0 & 0 \\ 0 & 1 & 0 \end{bmatrix}. \tag{17}$$

The hidden covariance matrix $\Sigma_h(s)$ has all elements zero, except the top-left most which is set to the innovation variance. To extract the first component of $h_t$ we use the (switch independent) $1 \times R$ projection matrix $B = \begin{bmatrix} 1 & 0 & \ldots & 0 \end{bmatrix}$. The (switch independent) visible scalar noise variance is given by $\Sigma_v \equiv \sigma_v^2$.

A well-known issue with raw speech signal models is that the energy of a signal may vary from one speaker to another or because of a change in recording conditions. For this reason the innovation $\Sigma_h$ is adjusted by maximizing the likelihood of an observed sequence with respect to the innovation covariance, a process called *Gain Adaptation* [16].

## 4.1 Training & Evaluation

Following [16], we trained a separate SAR-HMM for each of the eleven digits (0–9 and 'oh') from the TI-DIGITS database [17]. The training set for each digit was composed of 110 single digit utterances down-sampled to $8\,\mathrm{kHz}$, each one pronounced by a male speaker. Each SAR-HMM was composed of ten states with a left-right transition matrix. Each state was associated with a 10th-order AR process and the model was constrained to stay an integer multiple of $K = 140$ time steps (0.0175 seconds) in the same state. We refer the reader to [16] for a detailed explanation of the training procedure used with the SAR-HMM.

An AR-SLDS was built for each of the eleven digits by copying the parameters of the corresponding trained SAR-HMM, i.e., the AR coefficients $c_r(s)$ are copied into the first row of the hidden transition matrix $A(s)$ and the same discrete transition distribution $p(s_t \mid s_{t-1})$ is used. The models were then evaluated on a test set composed of 112 corrupted utterances of each of the eleven digits, each pronounced by different male speakers than those used in the training set. The recognition accuracy obtained by the models on the corrupted test sets is presented in Table 1. As expected, the performance of the SAR-HMM rapidly decreases with noise. The feature-based HMM with USS has high accuracy only for high SNR levels. In contrast, the AR-SLDS achieves a recognition accuracy of 61.2% at a SNR close to $0\,\mathrm{dB}$, while the performance of the two other methods is equivalent

to random guessing (9.1%). Whilst other inference methods may also perform well in this case, we found that EC performs admirably, without numerical instabilities, even for time-series with several thousand time-steps.

## 5   Discussion

We presented a method for approximate smoothed inference in an augmented class of switching linear dynamical systems. Our approximation is based on the idea that due to the forgetting which commonly occurs in Markovian models, a finite number of mixture components may provide a reasonable approximation. Clearly, in systems with very long correlation times our method may require too many mixture components to produce a satisfactory result, although we are unaware of other techniques that would be able to cope well in that case. The main benefit of EC over Kim smoothing is that future information is more accurately dealt with. Whilst EC is not as general as EP, EC carefully exploits the properties of singly-connected distributions, such as the aSLDS, to provide a numerically stable procedure. We hope that the ideas presented here may therefore help facilitate the practical application of dynamic hybrid networks.

### Acknowledgements

This work is supported by the EU Project FP6-0027787. This paper only reflects the authors' views and funding agencies are not liable for any use that may be made of the information contained herein.

## Footnotes

[1]$p(x|y)$ is a Gaussian with mean $\mu_x + \Sigma_{xy}\Sigma_{yy}^{-1}(y - \mu_y)$ and covariance $\Sigma_{xx} - \Sigma_{xy}\Sigma_{yy}^{-1}\Sigma_{yx}$.

[2]In the *exact* calculation, $p(h_{t+1}|s_t, s_{t+1}, v_{1:T})$ *is* a mixture of Gaussians, see [7]. However, since in (9) the two terms $p(h_{t+1}|s_{t+1}, v_{1:T})$ will only be approximately computed during the recursion, our approximation to $p(h_{t+1}|s_t, s_{t+1}, v_{1:T})$ will not be a mixture of Gaussians.

[3]This is a form of exact sampling since drawing samples from a Gaussian is easy. This should not be confused with meaning that this use of sampling renders EC a sequential Monte-Carlo scheme.

[4]Generalized EP [5], which groups variables together improves on the results, but is still far inferior to the EC results presented here – Onno Zoeter personal communication.

### References

[1]  Y. Bar-Shalom and Xiao-Rong Li. *Estimation and Tracking : Principles, Techniques and Software*. Artech House, Norwood, MA, 1998.

[2]  V. Pavlovic, J. M. Rehg, and J. MacCormick. Learning switching linear models of human motion. In *Advances in Neural Information Processing systems (NIPS 13)*, pages 981–987, 2001.

[3]  A. T. Cemgil, B. Kappen, and D. Barber. A Generative Model for Music Transcription. *IEEE Transactions on Audio, Speech and Language Processing*, 14(2):679 – 694, 2006.

[4]  U. N. Lerner. *Hybrid Bayesian Networks for Reasoning about Complex Systems*. PhD thesis, Stanford University, 2002.

[5]  O. Zoeter. *Monitoring non-linear and switching dynamical systems*. PhD thesis, Radboud University Nijmegen, 2005.

[6]  T. Minka. *A family of algorithms for approximate Bayesian inference*. PhD thesis, MIT Media Lab, 2001.

[7]  D. Barber. Expectation Correction for Smoothed Inference in Switching Linear Dynamical Systems. *Journal of Machine Learning Research*, 7:2515–2540, 2006.

[8]  C-J. Kim. Dynamic linear models with Markov-switching. *Journal of Econometrics*, 60:1–22, 1994.

[9]  T. Heskes and O. Zoeter. Expectation Propagation for approximate inference in dynamic Bayesian networks. In A. Darwiche and N. Friedman, editors, *Uncertainty in Art. Intelligence*, pages 216–223, 2002.

[10]  S. Lauritzen and F. Jensen. Stable local computation with conditional Gaussian distributions. *Statistics and Computing*, 11:191–203, 2001.

[11]  G. Kitagawa. The Two-Filter Formula for Smoothing and an implementation of the Gaussian-sum smoother. *Annals of the Institute of Statistical Mathematics*, 46(4):605–623, 1994.

[12]  Z. Ghahramani and G. E. Hinton. Variational learning for switching state-space models. *Neural Computation*, 12(4):963–996, 1998.

[13]  A. Doucet, N. de Freitas, and N. Gordon. *Sequential Monte Carlo Methods in Practice*. Springer, 2001.

[14]  B. Mesot and D. Barber. Switching Linear Dynamical Systems for Noise Robust Speech Recognition. IDIAP-RR 08, 2006.

[15]  G. Lathoud, M. Magimai-Doss, B. Mesot, and H. Bourlard. Unsupervised spectral subtraction for noise-robust ASR. In *Proceedings of ASRU 2005*, pages 189–194, November 2005.

[16]  Y. Ephraim and W. J. J. Roberts. Revisiting autoregressive hidden Markov modeling of speech signals. *IEEE Signal Processing Letters*, 12(2):166–169, February 2005.

[17]  R.G. Leonard. A database for speaker independent digit recognition. In *Proceedings of ICASSP84*, volume 3, 1984.
